# Threshold Learning for Optimal Decision Making

**Nathan F. Lepora**
Department of Engineering Mathematics, University of Bristol, UK
`n.lepora@bristol.ac.uk`

## Abstract

Decision making under uncertainty is commonly modelled as a process of competitive stochastic evidence accumulation to threshold (the drift-diffusion model). However, it is unknown how animals learn these decision thresholds. We examine threshold learning by constructing a reward function that averages over many trials to Wald's cost function that defines decision optimality. These rewards are highly stochastic and hence challenging to optimize, which we address in two ways: first, a simple two-factor reward-modulated learning rule derived from Williams' RE-INFORCE method for neural networks; and second, Bayesian optimization of the reward function with a Gaussian process. Bayesian optimization converges in fewer trials than REINFORCE but is slower computationally with greater variance. The REINFORCE method is also a better model of acquisition behaviour in animals and a similar learning rule has been proposed for modelling basal ganglia function.

## 1 Introduction

The standard view of perceptual decision making across psychology and neuroscience is of a competitive process that accumulates sensory evidence for the choices up to a threshold (bound) that triggers the decision [1, 2, 3]. While there is debate about whether humans and animals are 'optimal', nonetheless the standard psychological model of this process for two-alternative forced choices (the drift-diffusion model [1]) is a special case of an *optimal* statistical test for selecting between two hypotheses (the sequential probability ratio test, or SPRT [4]). Formally, this sequential test optimizes a cost function linear in the decision time and type I/II errors averaged over many trials [4]. Thus, under broad assumptions about the decision process, the optimal behaviour is simply to stop gathering data after reaching a threshold independent of the data history and collection time.

However, there remains the problem of how to set these decision thresholds. While there is consensus that an animal tunes its decision making by maximizing mean reward ([3, Chapter 5],[5, 6, 7, 8, 9, 10]), the learning rule is not known. More generally, it is unknown how an animal tunes its propensity towards making choices while also tuning its overall speed-accuracy balance.

Here we show that optimization of the decision thresholds can be considered as reinforcement learning over *single trial* rewards derived from Wald's *trial averaged* cost function considered previously. However, these single trial rewards are highly stochastic and their average has a broad flat peak (Fig. 1B), constituting a challenging optimization problem that will defeat standard methods. We address this challenge by proposing two distinct ways to learn the decision thresholds, with one approach closer to learning rules from neuroscience and the other to machine learning. The first approach is a learning rule derived from Williams' REINFORCE algorithm for training neural networks [11], which we here combine with an appropriate policy for controlling the thresholds for optimal decision making. The second is a Bayesian optimization method that fits a Gaussian process to the reward function and samples according to the mean reward and reward variance [12, 13, 14].

We find that both methods can successfully learn the thresholds, as validated by comparison against an exhaustive optimization of the reward function. Bayesian optimization converges in fewer trials

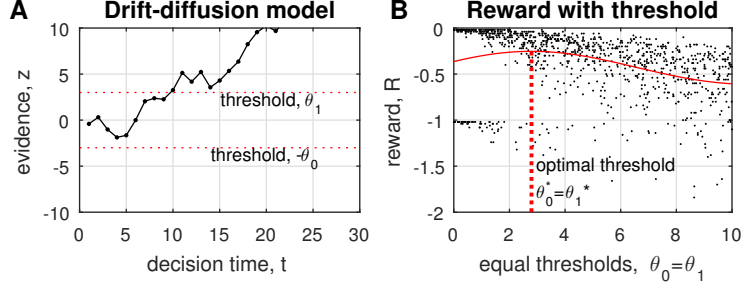

Figure 1: (A) Drift-diffusion model, representing a noisy stochastic accumulation until reaching a threshold when the decision is made. The optimal threshold maximizes the mean reward (equation 5). (B) Sampled rewards over 1000 trials with equal thresholds $\theta_0 = \theta_1$ (dotted markers); the average reward function is estimated from Gaussian process regression (red curve). Optimizing the thresholds is a challenging problem, particularly when the two thresholds are not equal.

($\sim 10^2$) than REINFORCE ($\sim 10^3$) but is 100-times more computationally expensive with about triple the variance in the threshold estimates. Initial validation is with one decision threshold, corresponding to equal costs of type I/II errors. The methods scale well to two thresholds (unequal costs), and we use REINFORCE to map the full decision performance over both costs. Finally, we compare both methods with experimental two-alternative forced choice data, and find that REINFORCE gives a better account of the acquisition (learning) phase, such as converging over a similar number of trials.

## 2 Background to the drift-diffusion model and SPRT

The drift-diffusion model (DDM) of Ratcliff and colleagues is a standard approach for modeling the results of two-alternative forced choice (2AFC) experiments in psychophysics [1, 15]. A decision variable $z(t)$ represents the sensory evidence accumulated to time $t$ from a starting bias $z(0) = z_0$. Discretizing time in uniform steps (assumed integer without losing generality), the update equation is

$$z(t + 1) = z(t) + \Delta z, \qquad \Delta z \sim N(\mu, \sigma^2), \tag{1}$$

where $\Delta$z is the increment of sensory evidence at time $t$, which is conventionally assumed drawn from a normal distribution $N(\mu, \sigma^2)$ of mean $\mu$ and variance $\sigma^2$. The decision criterion is that the accumulated evidence crosses one of two decision thresholds, assumed at $-\theta_0 < 0 < \theta_1$.

Wald's sequential probability ratio test (SPRT) optimally determines whether one of two hypotheses $H_0, H_1$ is supported by gathering samples $x(t)$ until a confident decision can be made [4]. It is optimal in that it minimizes the average sample size among all sequential tests to the same error probabilities. The SPRT can be derived from applying Bayes' rule recursively to sampled data, from when the log posterior ratio $\log \mathrm{PR}(t)$ passes one of two decision thresholds $-\theta_0 < 0 < \theta_1$:

$$\log \mathrm{PR}(t+1) = \log \mathrm{PR}(t) + \log \mathrm{LR}(t), \;\; \mathrm{PR}(t) = \frac{p(H_1|x(t))}{p(H_0|x(t))}, \;\; \mathrm{LR}(t) = \frac{p(x(t)|H_1)}{p(x(t)|H_0)}, \tag{2}$$

beginning from priors at time zero: $\mathrm{PR}(0) = p(H_1)/p(H_0)$. The right-hand side of equation (2) can also be written as a log likelihood ratio $\log \mathrm{LR}(t)$ summed over time $t$ (by iterative substitution).

The DDM is recognized as a special case of SPRT by setting the likelihoods as two equi-variant Gaussians $N(\mu_1, \sigma), N(\mu_0, \sigma)$, so that

$$\log \frac{p(x|H_1)}{p(x|H_0)} = \log \frac{e^{-(x-\mu_1)^2/2\sigma^2}}{e^{-(x-\mu_0)^2/2\sigma^2}} = \frac{\Delta\mu}{\sigma^2}x + d, \qquad \Delta\mu = \mu_1 - \mu_0, \; d = \frac{\mu_0^2 - \mu_1^2}{2\sigma^2}. \tag{3}$$

The integrated evidence $z(t)$ in (1) then coincides with the log posterior ratio in (2) and the increments $\Delta z$ with the log likelihood ratio in (2).

## 3 Methods to optimize the decision threshold

### 3.1 Reinforcement learning for optimal decision making

A general statement of decision optimality can be made in terms of minimizing the Bayes risk [4]. This cost function is linear in the type I and II error probabilities $\alpha_1 = P(H_1|H_0) = \mathbb{E}_1(e)$ and

$\alpha_0 = P(H_0|H_1) = \mathbb{E}_0(e)$, where the decision error $e = \{0,1\}$ for correct/incorrect trials, and is also linear in the expected stopping times for each decision outcome [1]

$$C_{\text{risk}} := \tfrac{1}{2}(W_0\alpha_0 + c\,\mathbb{E}_0[T]) + \tfrac{1}{2}(W_1\alpha_1 + c\,\mathbb{E}_1[T]), \qquad (4)$$

with type I/II error costs $W_0, W_1 > 0$ and cost of time $c$. That the Bayes risk $C_{\text{risk}}$ has a unique minimum follows from the error probabilities $\alpha_0$, $\alpha_1$ monotonically decreasing and the expected stopping times $\mathbb{E}_0[T], \mathbb{E}_1[T]$ monotonically increasing with increasing threshold $\theta_0$ or $\theta_1$. For each pair $(W_0/c, W_1/c)$, there is thus a unique threshold pair $(\theta_0^*, \theta_1^*)$ that minimizes $C_{\text{risk}}$.

We introduce reward into the formalism by supposing that an application of the SPRT with thresholds $(\theta_0, \theta_1)$ has a penalty proportional to the stopping time $T$ and decision outcome

$$R = \begin{cases} -W_0 - cT, & \text{incorrect decision of hypothesis } H_0 \\ -W_1 - cT, & \text{incorrect decision of hypothesis } H_1 \\ -cT, & \text{correct decision of hypothesis } H_0 \text{ or } H_1. \end{cases} \qquad (5)$$

Over many decision trials, the average reward is thus $\langle R \rangle = -C_{\text{risk}}$, the negative of the Bayes risk.

Reinforcement learning can then be used to find the optimal thresholds to maximize reward and thus optimize the Bayes risk. Over many trials $n = 1, 2, \ldots, N$ with reward $R(n)$, the problem is to estimate these optimal thresholds $(\theta_0^*, \theta_1^*)$ while maintaining minimal regret: the difference between the reward sum of the optimal decision policy and the sum of the collected rewards

$$\rho(N) = -NC_{\text{risk}}(\theta_0^*, \theta_1^*) - \sum_{n=1}^{N} R(n). \qquad (6)$$

This is recognized as a multi-armed bandit problem with a continuous two-dimensional action space parametrized by the threshold pairs $(\theta_0, \theta_1)$.

The optimization problem of finding the thresholds that maximize mean reward is highly challenging because of the stochastic decision times and errors. Standard approaches such as gradient ascent fail and even state-of-the-art approaches such as cross-entropy or natural evolution strategies are ineffective. A successful approach must combine reward averaging with learning (in a more sophisticated way than batch-averaging or filtering). We now consider two distinct approaches for this.

### 3.2 REINFORCE method

The first approach to optimize the decision threshold is a standard 2-factor learning rule derived from Williams' REINFORCE algorithm for training neural networks [11], but modified to the novel application of continuous bandits. From a modern perspective, the REINFORCE algorithm is seen as an example of a policy gradient method [16, 17]. These are well-suited to reinforcement learning with continuous action spaces, because they use gradient descent to optimize continuously parameterized policies with respect to cumulative reward.

We consider the decision thresholds $(\theta_0, \theta_1)$ to parametrize actions that correspond to making a single decision with those thresholds. Here we use a policy that expresses the threshold as a linear combination of binary unit outputs, with fixed coefficients specifying the contribution of each unit

$$\theta_0 = \sum_{j=1}^{n_s} s_j y_j, \qquad \theta_1 = \sum_{j=n_s+1}^{2n_s} s_j y_j. \qquad (7)$$

Exponential coefficients were found to work well (equivalent to binary encoding), scaled to give a range of thresholds from zero to $\theta_{\max}$:

$$s_j = s_{n_s+j} = \frac{(1/2)^j}{1 - (1/2)^{n_s}} \theta_{\max}, \qquad (8)$$

where here we use $n_s = 10$ units per threshold with maximum threshold $\theta_{\max} = 10$. The benefit of this policy (7,8) is that the learning rule can be expressed in terms of the binary unit outputs $y_j = \{0,1\}$, which are the variables considered in the REINFORCE learning rule [11].

Following Williams, the policy choosing the threshold on a trial is stochastic by virtue of the binary unit outputs $y_j = \{0,1\}$ being distributed according to a logistic function of weights $w_j$, such that

$$y_j \sim p(y_j|w_j) = f(w_j)y_j + (1 - f(w_j))(1 - y_j), \quad f(w_j) = \frac{1}{1 + e^{-w_j}}. \qquad (9)$$

The REINFORCE learning rule for these weights is determined by the reward $R(n)$ on trial $n$

$$\Delta w_j = \beta \left[ y_j(t) - f(w_j) \right] R(n), \tag{10}$$

with learning rate $\beta$ (here generally taken as 0.1). An improvement to the learning rule can be made with reinforcement comparison, with a reference reward $\bar{R}(n) = \gamma R(n) + (1 - \gamma) \bar{R}(n-1)$ subtracted from $R(n)$; a value $\gamma = 0.5$ was found to be effective, and is used in all simulations using the REINFORCE rule in this paper.

The power of the REINFORCE learning rule is that the weight change is equal to the gradient of the expected return $J(\boldsymbol{w}) = \mathbb{E}[R_{\{\theta\}}]$ over all possible threshold sequences $\{\theta\}$. Thus, a single-trial learning rule performs like stochastic gradient ascent averaged over many trials. Note also that the neural network input $x_i$ of the original formalism [11] is here set to $x_1 = 1$, but a non-trivial input could be used to aid learning recall and generalization (see discussion). Overall, the learning follows a reward-modulated two-factor rule that recruits units distributed according to an exponential size principle, and thus resembles models of biological motor learning.

### 3.3   Bayesian optimization method

The second approach is to use Bayesian optimization to find the optimal thresholds from iteratively building a probabilistic model of the reward function that is used to guide future sampling [12, 13, 14]. Bayesian optimization typically uses a Gaussian process model, which provides a nonlinear regression model both of the mean reward and the reward variance with decision threshold. This model can then be used to guide future threshold choice via maximising an acquisition function of these quantities.

The basic algorithm for Bayesian optimization is as follows:

---
**Algorithm**  Bayesian optimization applied to optimal decision making
---
**for** n=1 to $N$ **do**
    New thresholds from optimizing acquisition function $(\theta_0, \theta_1)_n = \underset{(\theta_0, \theta_1)}{\operatorname{argmax}} \, \alpha(\theta_0, \theta_1; \mathcal{D}_{n-1})$
    Make the decision with thresholds $(\theta_0, \theta_1)_n$ to find reward $R(n)$
    Augment data by including new samples $\mathcal{D}_n = (\mathcal{D}_{n-1}; (\theta_0, \theta_1)_n, R(n))$
    Update the statistical (Gaussian process) model of the rewards
**end for**
---

Following other work on Bayesian optimization, we model the reward dependence on the decision thresholds with a Gaussian process

$$R(\theta_0, \theta_1) \sim \mathcal{GP}[m(\theta_0, \theta_1), k(\theta_0, \theta_1; \theta_0', \theta_1')], \tag{11}$$

with mean $m(\theta_0, \theta_1) = \mathbb{E}[R(\theta_0, \theta_1)]$ and covariance modelled by a squared-exponential function

$$k(\theta_0, \theta_1; \theta_0', \theta_1') = \sigma_f^2 \exp\left( -\tfrac{\lambda}{2} ||(\theta_0, \theta_1) - (\theta_0', \theta_1')||^2 \right). \tag{12}$$

The fitting of the hyperparameters $\sigma_f^2$, $\lambda$ used standard methods [18] (GPML toolbox and a quasi-Newton optimizer in MATLAB). In principle, the two thresholds could each have distinct hyperparameters, but we use one to maintain the symmetry $\theta_0 \leftrightarrow \theta_1$ of the decision problem.

The choice of decision thresholds is viewed as a sampling problem, and represented by maximizing an acquisition function of the decision thresholds that trades off exploration and exploitation. Here we use the probability of improvement, which guides the sampling towards regions of high uncertainty and reward by maximizing the chance of improving the present best estimate:

$$(\theta_0, \theta_1)_n = \underset{(\theta_0, \theta_1)}{\operatorname{argmax}} \, \alpha(\theta_0, \theta_1), \qquad \alpha(\theta_0, \theta_1) = \Phi\left( \frac{m(\theta_0, \theta_1) - R(\theta_0^*, \theta_1^*)}{k(\theta_0, \theta_1; \theta_0, \theta_1)} \right), \tag{13}$$

where $(\theta_0^*, \theta_1^*)$ are the threshold estimates that have given the greatest reward and $\Phi$ is the normal cumulative distribution function. Usually one would include a noise parameter for exploration, but because the decision making is stochastic we use the noise from that process instead.

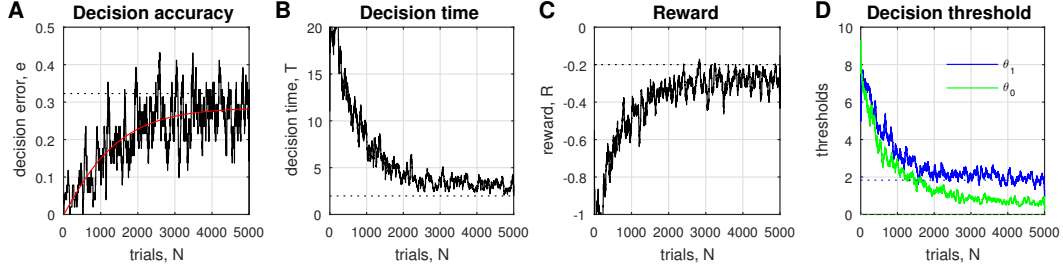

Figure 2: REINFORCE learning (exponential coefficients) of the two decision thresholds over a single learning episode. Decision costs $c = 0.05$, $W_0 = 0.1$ and $W_1 = 1$. Plots are smoothed over 50 trials. The red curve is the average accuracy by trial number (fitted to a cumulative Weibull function). Optimal values (from exhaustive optimization) are shown as dashed lines.

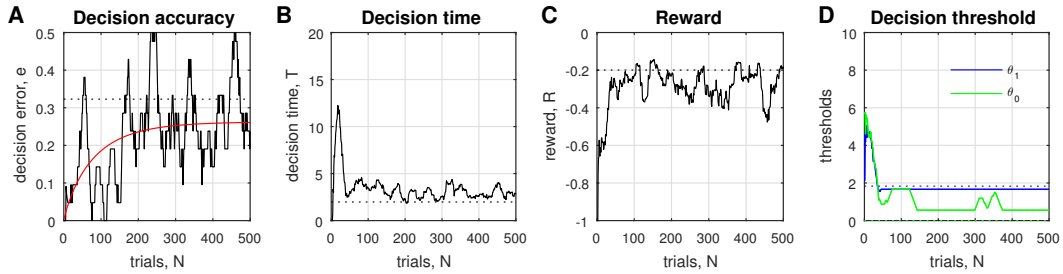

Figure 3: Bayesian optimization of the two decision thresholds over a single learning episode. Other details are the same as in Fig. 2, other than only 500 trials were used with smoothing over 20 trials.

## 4 Results

### 4.1 Single learning episode

The learning problem is to find the pair of optimal decision thresholds $(\theta_0^*, \theta_1^*)$ that maximize the reward function (5), which is a linear combination of penalties for delays and type I and II errors. The reward function has two free parameters that affect the optimal thresholds: the costs $W_0/c$ and $W_1/c$ of making type I and II errors relative to time. The methods apply generally, although for concreteness we consider a drift-diffusion model equivalent to the SPRT with distribution means $\mu_0 = -\mu_1 = 1/3$ and standard deviation $\sigma = 1$.

Both the REINFORCE method and Bayesian optimization can converge to approximations of the optimal decision thresholds, as shown in Figures 2D,3D above for a typical learning episode. The decision error $e$, decision time $T$ and reward $R$ are all highly variable from the stochastic nature of the evidence, although displayed plots have their variance reduced by smoothing over 50 trials (to help interpret the results). There is a gradual convergence towards near optimal decision performance.

Clearly the main difference between the REINFORCE method and the Bayesian optimization method is the speed of convergence to the decision thresholds (*c.f.* Figures 2D *vs* 3D). REINFORCE gradually converges over ~5000 trials whereas Bayesian optimization converges in $\lesssim 500$ trials. However, there are other differences between the two methods that are only revealed for multiple learning episodes, which act to balance the pros and cons across the two methods.

### 4.2 Multiple learning episodes: one decision threshold

For validation purposes, we reduce the learning problem to the simpler case where there is only one decision threshold $\theta_0 = \theta_1$, by setting costs equal for type I and II errors $W_0/c = W_1/c$ so that the error probabilities are equal $\alpha_0 = \alpha_1$. This will allow us to compare the two methods in a representative scenario that is simpler to visualize and can be validated against an exhaustive optimization of the reward function (which takes too long to calculate for two thresholds).

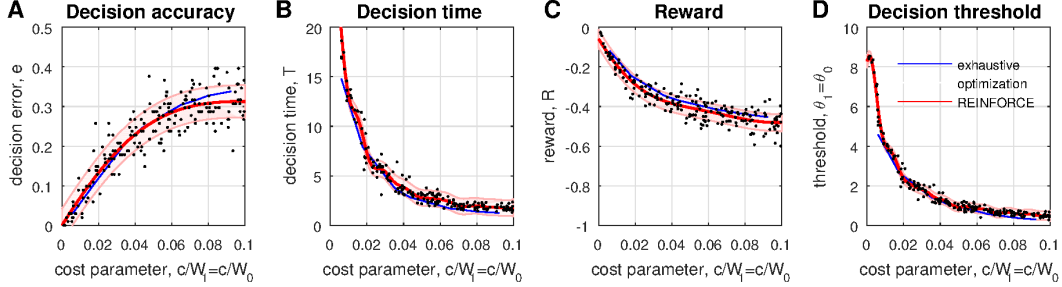

Figure 4: REINFORCE learning of one decision threshold (for equal thresholds $\theta_1 = \theta_0$) over 200 learning episodes with costs $c/W_1 = c/W_0$ sampled uniformly from $[0, 0.1]$. Results are after 5000 learning trials (averaged over 100 trials). The mean and standard deviation of these results (red line and shaded region) are compared with an exhaustive optimization over $10^6$ episodes (blue curves).

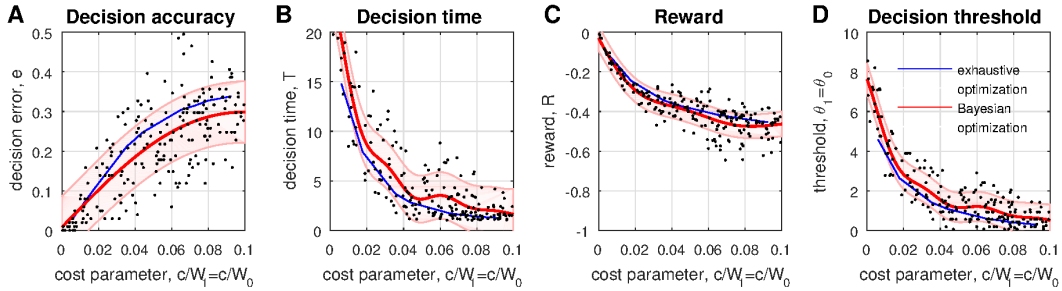

Figure 5: Bayesian optimization of one decision threshold (for equal thresholds $\theta_1 = \theta_0$) over 200 learning episodes with costs $c/W_1 = c/W_0$ sampled uniformly from $[0, 0.1]$. Results are after 500 learning trials (averaged over 100 trials). The mean and standard deviation of these results (red line and shaded region) are compared with an exhaustive optimization over $10^6$ episodes (blue curves).

We consider REINFORCE over 5000 trials and Bayesian optimization over 500 trials, which are sufficient for convergence (Figures 2,3). Costs were considered over a range $W/c > 10$ via random uniform sampling of $c/W$ over the range $[0, 0.1]$. Mean decision errors $e$, decision times $T$, rewards and thresholds are averaged over the final 50 trials, combining the results for both choices.

Both the REINFORCE and Bayesian optimization methods estimate near-optimal decision thresholds for all considered cost parameters (Figures 4,5; red curves) as verified from comparison with an exhaustive search of the reward function (blue curves) over $10^6$ decision trials (randomly sampling the threshold range to estimate an average reward function, as in Fig 1B). In both cases, the exhaustive search lies within one standard deviation of the decision threshold from the two learning methods.

There are, however, differences in performance between the two methods. Firstly, the variance of the threshold estimates is greater for Bayesian optimization than for REINFORCE (*c.f.* Figures 4D vs 5D). The variance of the decision thresholds feeds through into larger variances for the decision error, time and reward. Secondly, although Bayesian optimization converges in fewer trials (500 *vs* 5000), it comes at the expense of greater computational cost of the algorithm (Table 1).

The above results were checked for robustness across reasonable ranges of the various metaparameters for each learning method. For REINFORCE, the results were not appreciably affected by having any learning rate $\beta$ within the range 0.1-1; similarly, increasing the unit number $n$ did not affect the threshold variances, but scales the computation time.

### 4.3 Multiple learning episodes: two decision thresholds

We now consider the learning problem with two decision thresholds $(\theta_0, \theta_1)$ that optimize the reward function 5 with differing $W_0/c$ and $W_1/c$ values. We saw above that REINFORCE produces the more accurate estimates relative to the computational cost, so we concentrate on that method only.

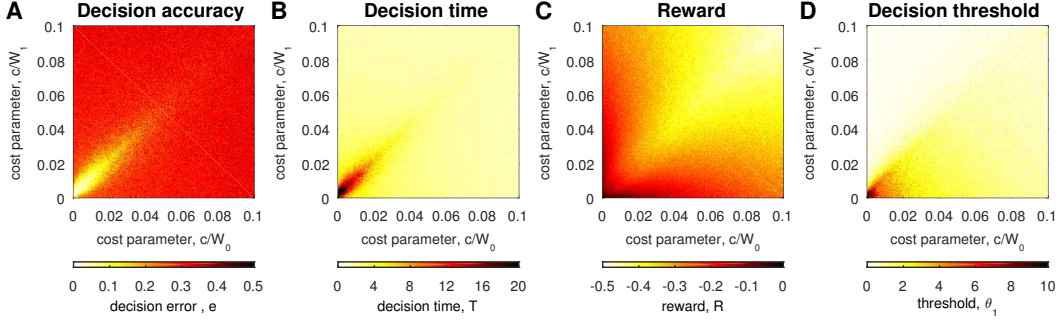

Figure 6: Reinforcement learning of two decision thresholds. Method same as Figure 4 except that $200^2$ learning episodes are considered with costs $(c/W_0, c/W_1)$ sampled from $[0, 0.1] \times [0, 0.1]$. The threshold $\theta_0$ results are just reflections of those for $\theta_1$ in the axis $c/W_0 \leftrightarrow c/W_1$ and thus not shown.

Table 1: Comparison of threshold learning methods. Results for one decision threshold, averaging over the data in Figures 4,5. (Benchmarked on an i7 2.7GHz CPU.)

|  | REINFORCE method | Bayesian optimization | Exhaustive optimization |
|---|---|---|---|
| computation time | 0.5 sec (5000 trials) | 50 sec (500 trials) | 44 sec ($10^6$ trials) |
| computation time/trial | 0.1 msec/trial | 100 msec/trial | 0.04 msec/trial |
| uncertainty, $\Delta\theta$ (1 s.d.) | 0.23 | 0.75 | 0.01 |

The REINFORCE method can find the two decision thresholds (Figure 6), as demonstrated by estimating the thresholds over $200^2$ instances of the reward function with $(c/W_0, c/W_1)$ sampled uniformly from $[0, 0.1] \times [0, 0.1]$. Because of the high compute time, we cannot compare the results to those from an exhaustive search, apart from that the plot diagonals ($W_0/c = W_1/c$) reduce to the single threshold results which matched an exhaustive optimization (Figure 4).

Figure 6 is of general interest because it maps the drift-diffusion model (SPRT) decision performance over a main portion of its parameter space. Results for the two decision thresholds $(\theta_0, \theta_1)$ are reflections of each other about $W_0 \leftrightarrow W_1$, while the decision error, time and reward are reflection symmetric (consistent with these symmetries of the decision problem). All quantities depend on both weight parameters $(W_0/c, W_1/c)$ in a smooth but non-trivial manner. To our knowledge, this is the first time the full decision performance has been mapped.

## 4.4 Comparison with animal learning

The relation between reward and decision optimality is directly relevant to the psychophysics of two alternative forced choice tasks in the tradeoff between decision accuracy and speed [3]. Multiple studies support that the decision threshold is set to maximize reward [7, 8, 9]. However, the mechanism by which subjects learn the optimal thresholds has not been addressed. Our two learning methods are candidate mechanisms, and thus should be compared with experiment.

We have found a couple of studies showing data over the acquisition phase of two-alternative forced choice behavioural experiments: one for rodent whisker vibrotactile discrimination [19, Figure 4] and the other for bat echoacoustic discrimination [20]. Studies detailing the acquisition phase are rare compared to those of the proficient phase, even though they are a necessary component of all such behavioural experiments (and successful studies rest on having a well-designed acquisition phase).

In both behavioural studies, the animals acquired proficient decision performance after 5000-10000 trials: in rodent, this was after 25-50 sessions of $\sim$200 trials [19, Figure 4]; and in bat, after about 6000 trials for naive animals [20, Figure 4]. The typical progress of learning was to begin with random choices (mean decision error $e = 0.5$) and then gradually converge on the appropriate balance of decision time *vs* accuracy. There was considerable variance in final performance across different animals (in rodent, mean decision errors were $e \sim 0.05$-$0.15$).

That acquisition takes 5000 or more trials is consistent with the REINFORCE learning rule (Figure 2), and not with Bayesian optimization (Figure 3). Moreover, the shape of the acquisition curve for the REINFORCE method resembles that of the animal learning, in also having a good fit to a cumulative Weibull function over a similar number of trials (red line, Figure 2). That being said, the animals begin making random choices and gradually improve in accuracy with longer decision times, whereas both artificial learning methods (Figures 2,3) begin with accurate choices and then decrease in accuracy and decision time. Taken together, this evidence supports that the REINFORCE learning rule is a plausible model of animal learning, although further theoretical and experimental study is required.

## 5  Discussion

We examined how to learn decision thresholds in the drift-diffusion model of perceptual decision making. A key step was to use *single trial* rewards derived from Wald's trial-averaged cost function for the equivalent sequential probability ratio test, which took the simple form of a linear weighting of penalties due to time and type I/II errors. These highly stochastic rewards are challenging to optimize, which we addressed with two distinct methods to learn the decision thresholds.

The first approach for learning the thresholds was based on a method for training neural networks known as Williams' REINFORCE rule [11]. In modern terminology, this can be viewed as a policy gradient method [16, 17] and here we proposed an appropriate policy for optimal decision making. The second method was a modern Bayesian optimization method that samples and builds a probabilistic model of the reward function to guide further sampling [12, 13, 14]. Both learning methods converged to nearby the optimum decision thresholds, as validated against an exhaustive optimization (over $10^6$ trials). The Bayesian optimization method converged much faster ($\sim$500 trials) than the REINFORCE method ($\sim$5000 trials). However, Bayesian optimization is three-times as variable in the threshold estimates and 40-times slower in computation time. It appears that the faster convergence for Bayesian optimization leads to less averaging over the stochastic rewards, and hence greater variance than with the REINFORCE method.

We expect that both the REINFORCE and Bayesian optimization methods used here can be improved to compensate for some of their individual drawbacks. For example, the full REINFORCE learning rule has a third factor corresponding to the neural network input, which could represent a context signal to allow recall and generalization over past learnt thresholds; also, information on past trial performance is discarded by REINFORCE, which could be partially retained to improve learning. Bayesian optimization could be improved in computational speed by updating the Gaussian process with just the new samples after each decision, rather than refitting the entire Gaussian process; also, the variance of the threshold estimates may improve with other choices of acquisition function for sampling the rewards or other assumptions for the Gaussian process covariance function. In addition, the optimization methods may have broader applicability when the optimal decision thresholds vary with time [10], such as tasks with deadlines or when there are multiple (three or more) choices.

Several more factors support the REINFORCE method as a model of reward-driven learning during perceptual decision making. First, REINFORCE is based on a neural network and is thus better suited as a connectionist model of brain function. Second, the REINFORCE model results (Fig. 2) resemble acquisition data from behavioural experiments in rodent [19] and bat [20] (Sec. 4.4). Third, the site of reward learning would plausibly be the basal ganglia, and a similar 3-factor learning rule has already been used to model cortico-striatal plasticity [21]. In addition, multi-alternative (MSPRT) versions of the drift-diffusion model offer a model of action selection in the basal ganglia [22, 23], and so the present REINFORCE model of decision acquisition would extend naturally to encompass a combined model of reinforcement learning and optimal decision making in the brain.

**Acknowledgements**

I thank Jack Crago, John Lloyd, Kirsty Aquilina, Kevin Gurney and Giovanni Pezzulo for discussions related to this research. The code used to generate the results and figures for this paper is at http://lepora.com/publications.htm

## Footnotes

[1] The full expression has prior probabilities for the frequency of each outcome, which are here assumed equal.

## References

[1] R. Ratcliff. A theory of memory retrieval. *Psychological Review*, 85:59–108, 1978.

[2] J. Gold and M. Shadlen. The neural basis of decision making. *Annu. Rev. Neurosci.*, 30:535–574, 2007.

[3] R. Bogacz, E. Brown, J. Moehlis, P. Holmes, and J.D. Cohen. The physics of optimal decision making: A formal analysis of models of performance in two-alternative forced-choice tasks. *Psychological Review*, 113(4):700, 2006.

[4] A. Wald and J. Wolfowitz. Optimum character of the sequential probability ratio test. *The Annals of Mathematical Statistics*, 19(3):326–339, 1948.

[5] J. Gold and M. Shadlen. Banburismus and the brain: decoding the relationship between sensory stimuli, decisions, and reward. *Neuron*, 36(2):299–308, 2002.

[6] P. Simen, J. Cohen, and P. Holmes. Rapid decision threshold modulation by reward rate in a neural network. *Neural networks*, 19(8):1013–1026, 2006.

[7] P. Simen, D. Contreras, C. Buck, P. Hu, and J. Holmes, P.and Cohen. Reward rate optimization in two-alternative decision making: empirical tests of theoretical predictions. *Journal of Experimental Psychology: Human Perception and Performance*, 35(6):1865, 2009.

[8] R. Bogacz, P. Hu, P. Holmes, and J. Cohen. Do humans produce the speed–accuracy trade-off that maximizes reward rate? *The Quarterly Journal of Experimental Psychology*, 63(5):863–891, 2010.

[9] F. Balci, P. Simen, R. Niyogi, A. Saxe, J. Hughes, P. Holmes, and J. Cohen. Acquisition of decision making criteria: reward rate ultimately beats accuracy. *Attention, Perception, & Psychophysics*, 73(2):640–657, 2011.

[10] J. Drugowitsch, R. Moreno-Bote, A. Churchland, M. Shadlen, and A. Pouget. The cost of accumulating evidence in perceptual decision making. *The Journal of Neuroscience*, 32(11):3612–3628, 2012.

[11] R. Williams. Simple statistical gradient-following algorithms for connectionist reinforcement learning. *Machine learning*, 8(3-4):229–256, 1992.

[12] M. Pelikan. Bayesian optimization algorithm. In *Hierarchical Bayesian optimization algorithm*, pages 31–48. Springer, 2005.

[13] E. Brochu, V. Cora, and N. De Freitas. A tutorial on bayesian optimization of expensive cost functions, with application to active user modeling and hierarchical reinforcement learning. *arXiv preprint arXiv:1012.2599*, 2010.

[14] J. Snoek, H. Larochelle, and R. Adams. Practical bayesian optimization of machine learning algorithms. In *Advances in neural information processing systems*, pages 2951–2959, 2012.

[15] R. Ratcliff and G. McKoon. The diffusion decision model: theory and data for two-choice decision tasks. *Neural computation*, 20(4):873–922, 2008.

[16] J. Peters and S. Schaal. Reinforcement learning of motor skills with policy gradients. *Neural networks*, 21(4):682–697, 2008.

[17] R. Sutton, D. McAllester, S. Singh, and Y. Mansour. Policy gradient methods for reinforcement learning with function approximation. In *Neural Information Processing Systems 12*, pages 1057–1063, 2000.

[18] C. Rasmussen and C. Williams. *Gaussian Processes for Machine Learning.* the MIT Press, 2006.

[19] J. Mayrhofer, V. Skreb, W. von der Behrens, S. Musall, B. Weber, and F. Haiss. Novel two-alternative forced choice paradigm for bilateral vibrotactile whisker frequency discrimination in head-fixed mice and rats. *Journal of neurophysiology*, 109(1):273–284, 2013.

[20] K. Stich and Y. Winter. Lack of generalization of object discrimination between spatial contexts by a bat. *Journal of experimental biology*, 209(23):4802–4808, 2006.

[21] M. Frank and E. Claus. Anatomy of a decision: striato-orbitofrontal interactions in reinforcement learning, decision making, and reversal. *Psychological review*, 113(2):300, 2006.

[22] R. Bogacz and K. Gurney. The basal ganglia and cortex implement optimal decision making between alternative actions. *Neural computation*, 19(2):442–477, 2007.

[23] N. Lepora and K. Gurney. The basal ganglia optimize decision making over general perceptual hypotheses. *Neural Computation*, 24(11):2924–2945, 2012.

